# A Neural Edge-Detection Model for Enhanced Auditory Sensitivity in Modulated Noise

**Alon Fishbach and Bradford J. May**
Department of Biomedical Engineering and Otolaryngology-HNS
Johns Hopkins University
Baltimore, MD 21205
*fishbach@northwestern.edu*

## Abstract

Psychophysical data suggest that temporal modulations of stimulus amplitude envelopes play a prominent role in the perceptual segregation of concurrent sounds. In particular, the detection of an unmodulated signal can be significantly improved by adding amplitude modulation to the spectral envelope of a competing masking noise. This perceptual phenomenon is known as "Comodulation Masking Release" (CMR). Despite the obvious influence of temporal structure on the perception of complex auditory scenes, the physiological mechanisms that contribute to CMR and auditory streaming are not well known. A recent physiological study by Nelken and colleagues has demonstrated an enhanced cortical representation of auditory signals in modulated noise. Our study evaluates these CMR-like response patterns from the perspective of a hypothetical auditory edge-detection neuron. It is shown that this simple neural model for the detection of amplitude transients can reproduce not only the physiological data of Nelken et al., but also, in light of previous results, a variety of physiological and psychoacoustical phenomena that are related to the perceptual segregation of concurrent sounds.

## 1 Introduction

The temporal structure of a complex sound exerts strong influences on auditory physiology (e.g. [10, 16]) and perception (e.g. [9, 19, 20]). In particular, studies of auditory scene analysis have demonstrated the importance of the temporal structure of amplitude envelopes in the perceptual segregation of concurrent sounds [2, 7]. Common amplitude transitions across frequency serve as salient cues for grouping sound energy into unified perceptual objects. Conversely, asynchronous amplitude transitions enhance the separation of competing acoustic events [3, 4].

These general principles are manifested in perceptual phenomena as diverse as comodulation masking release (CMR) [13], modulation detection interference [22] and synchronous onset grouping [8].

Despite the obvious importance of timing information in psychoacoustic studies of auditory masking, the way in which the CNS represents the temporal structure of an amplitude envelope is not well understood. Certainly many physiological studies have demonstrated neural sensitivities to envelope transitions, but this sensitivity is only beginning to be related to the variety of perceptual experiences that are evoked by signals in noise.

Nelken et al. [15] have suggested a correspondence between neural responses to time-varying amplitude envelopes and psychoacoustic masking phenomena. In their study of neurons in primary auditory cortex (A1), adding temporal modulation to background noise lowered the detection thresholds of unmodulated tones. This enhanced signal detection is similar to the perceptual phenomenon that is known as comodulation masking release [13].

Fishbach et al. [11] have recently proposed a neural model for the detection of "auditory edges" (i.e., amplitude transients) that can account for numerous physiological [14, 17, 18] and psychoacoustical [3, 21] phenomena. The encompassing utility of this edge-detection model suggests a common mechanism that may link the auditory processing and perception of auditory signals in a complex auditory scene. Here, it is shown that the auditory edge detection model can accurately reproduce the cortical CMR-like responses previously described by Nelken and colleagues.

## 2 The Model

The model is described in detail elsewhere [11]. In short, the basic operation of the model is the calculation of the first-order time derivative of the log-compressed envelope of the stimulus. A computational model [23] is used to convert the acoustic waveform to a physiologically plausible auditory nerve representation (Fig 1$a$). The simulated neural response has a medium spontaneous rate and a characteristic frequency that is set to the frequency of the target tone. To allow computation of the time derivative of the stimulus envelope, we hypothesize the existence of a temporal delay dimension, along which the stimulus is progressively delayed. The intermediate delay layer (Fig 1$b$) is constructed from an array of neurons with ascending membrane time constants ($\tau$); each neuron is modeled by a conventional integrate-and-fire model (I&F, [12]). Higher membrane time constant induces greater delay in the neuron's response [1].

The output of the delay layer converges to a single output neuron (Fig. 1$c$) via a set of connection with various efficacies that reflect a receptive field of a gaussian derivative. This combination of excitatory and inhibitory connections carries out the time-derivative computation. Implementation details and parameters are given in [11]. The model has 2 adjustable and 6 fixed parameters, the former were used to fit the responses of the model to single unit responses to variety of stimuli [11]. The results reported here are not sensitive to these parameters.

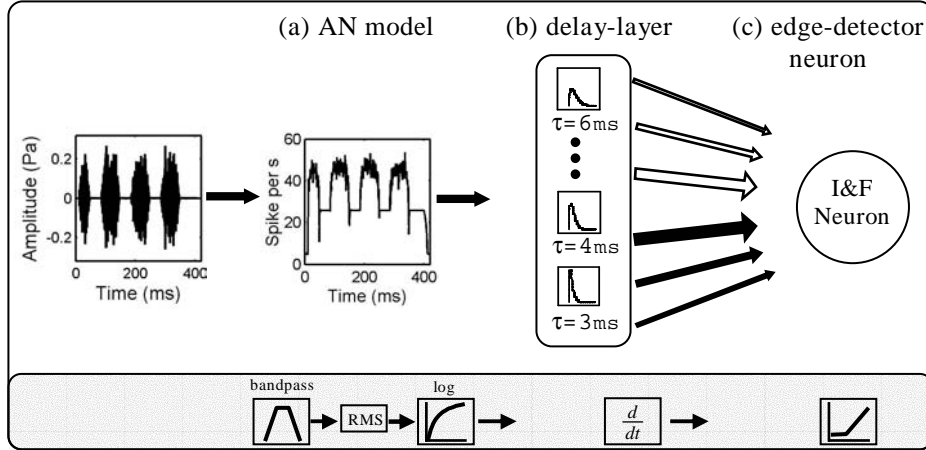

Figure 1: Schematic diagram of the model and a block diagram of the basic operation of each model component (shaded area). The stimulus is converted to a neural representation (*a*) that approximates the average firing rate of a medium spontaneous-rate AN fiber [23]. The operation of this stage can be roughly described as the log-compressed rms output of a bandpass filter. The neural representation is fed to a series of neurons with ascending membrane time constant (*b*). The kernel functions that are used to simulate these neurons are plotted for a few neurons along with the time constants used. The output of the delay-layer neurons converge to a single I&F neuron (*c*) using a set of connections with weights that reflect a shape of a gaussian derivative. Solid arrows represent excitatory connections and white arrows represent inhibitory connections. The absolute efficacy is represented by the width of the arrows.

## 3   Results

Nelken et al. [15] report that amplitude modulation can substantially modify the noise-driven discharge rates of A1 neurons in Halothane-anesthetized cats. Many cortical neurons show only a transient onset response to unmodulated noise but fire in synchrony ("lock") to the envelope of modulated noise. A significant reduction in envelope-locked discharge rates is observed if an unmodulated tone is added to modulated noise. As summarized in Fig. 2, this suppression of envelope locking can reveal the presence of an auditory signal at sound pressure levels that are not detectable in unmodulated noise. It has been suggested that this pattern of neural responding may represent a physiological equivalent of CMR.

Reproduction of CMR-like cortical activity can be illustrated by a simplified case in which the analytical amplitude envelope of the stimulus is used as the input to the edge-detector model. In keeping with the actual physiological approach of Nelken et al., the noise envelope is shaped by a trapezoid modulator for these simulations. Each cycle of modulation, $E_N(t)$, is given by:

$$E_N(t) = \begin{cases} \frac{P}{D}t & 0 \le t < D \\ P & D \le t < 3D \\ P - \frac{P}{D}(t - 3D) & 3D \le t < 4D \\ 0 & 4D \le t < 8D \end{cases}$$

where *P* is the peak pressure level and *D* is set to 12.5 ms.

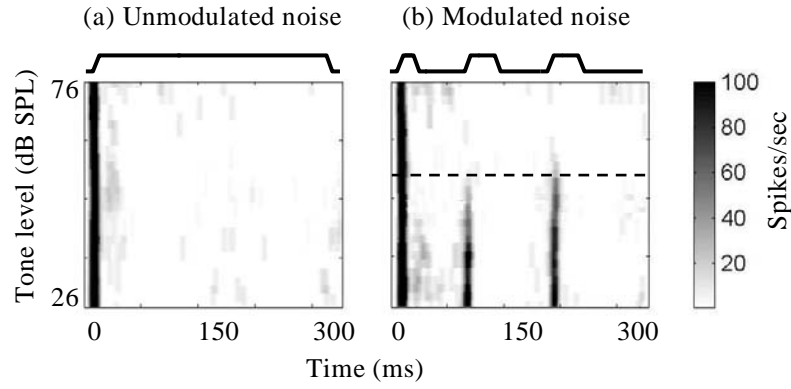

Figure 2: Responses of an A1 unit to a combination of noise and tone at many tone levels, replotted from Nelken et al. [15]. (*a*) Unmodulated noise and (*b*) modulated noise. The noise envelope is illustrated by the thick line above each figure. Each row shows the response of the neuron to the noise plus the tone at the level specified on the ordinate. The dashed line in (*b*) indicates the detection threshold level for the tone. The detection threshold (as defined and calculated by Nelken et al.) in the unmodulated noise was not reached.

Since the basic operation of the model is the calculation of the rectified time-derivative of the log-compressed envelope of the stimulus, the expected noise-driven rate of the model can be approximated by:

$$M_N(t) = \max\left\{0, \frac{d}{dt} A \ln\left(1 + \frac{E(t)}{P_0}\right)\right\}$$

where $A = 20/\ln(10)$ and $P_0 = 2e\text{-}5$ Pa. The expected firing rate in response to the noise plus an unmodulated signal (tone) can be similarly approximated by:

$$M_{N+S}(t) = \max\left\{0, \frac{d}{dt} A \ln\left(1 + \frac{E(t) + P_S}{P_0}\right)\right\}$$

where $P_S$ is the peak pressure level of the tone. Clearly, both $M_N(t)$ and $M_{N+S}(t)$ are identically zero outside the interval [0 $D$]. Within this interval it holds that:

$$M_N(t) = \frac{\frac{AP}{D}}{P_0 + \frac{P}{D}t} \quad 0 \le t < D \qquad \text{and} \qquad M_{N+S}(t) = \frac{\frac{AP}{D}}{P_0 + P_S + \frac{P}{D}t} \quad 0 \le t < D$$

and the ratio of the firing rates is:

$$\frac{M_N(t)}{M_{N+S}(t)} = 1 + \frac{P_S}{P_0 + \frac{P}{D}t} \quad 0 \le t < D$$

Clearly, $M_{N+S} < M_N$ for the interval [0 $D$] of each modulation cycle. That is, the addition of a tone reduces the responses of the model to the rising part of the modulated envelope. Higher tone levels ($P_s$) cause greater reduction in the model's firing rate.

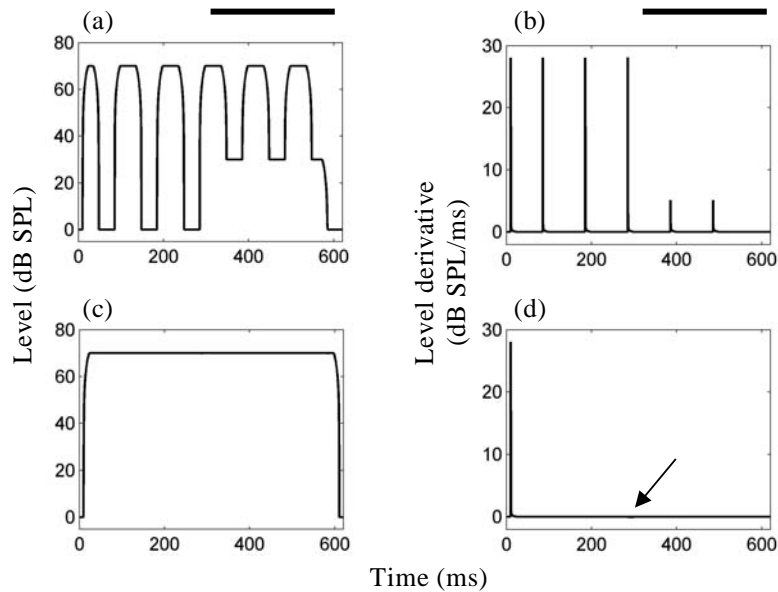

Figure 3: An illustration of the basic operation of the model on various amplitude envelopes. The simplified operation of the model includes log compression of the amplitude envelope (*a* and *c*) and rectified time-derivative of the log-compressed envelope (*b* and *d*). (*a*) A 30 dB SPL tone is added to a modulated envelope (peak level of 70 dB SPL) 300 ms after the beginning of the stimulus (as indicated by the horizontal line). The addition of the tone causes a great reduction in the time derivative of the log-compressed envelope (*b*). When the envelope of the noise is unmodulated (*c*), the time-derivative of the log-compressed envelope (*d*) shows a tiny spike when the tone is added (marked by the arrow).

Fig. 3 demonstrates the effect of a low-level tone on the time-derivative of the log-compressed envelope of a noise. When the envelope is modulated (Fig. 3*a*) the addition of the tone greatly reduces the derivative of the rising part of the modulation (Fig. 3*b*). In the absence of modulations (Fig. 3*c*), the tone presentation produces a negligible effect on the level derivative (Fig. 3*d*).

Model simulations of neural responses to the stimuli used by Nelken et al. are plotted in Fig. 4. As illustrated schematically in Fig 3 (*d*), the presence of the tone does not cause any significant change in the responses of the model to the unmodulated noise (Fig. 4*a*). In the modulated noise, however, tones of relatively low levels reduce the responses of the model to the rising part of the envelope modulations.

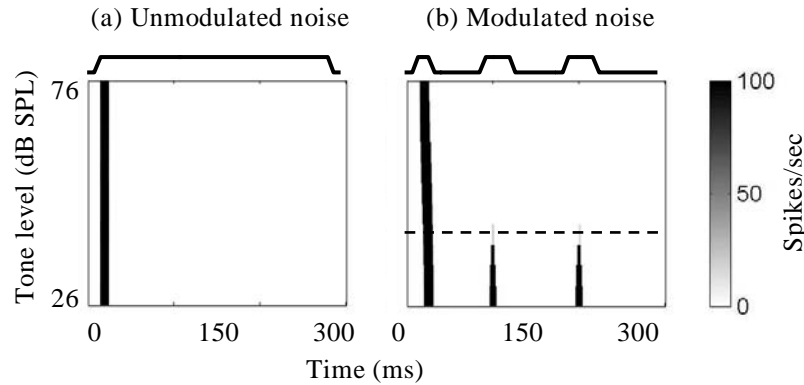

Figure 4: Simulated responses of the model to a combination of a tone and Unmodulated noise (*a*) and modulated noise (*b*). All conventions are as in Fig. 2.

## 4  Discussion

This report uses an auditory edge-detection model to simulate the actual physiological consequences of amplitude modulation on neural sensitivity in cortical area A1. The basic computational operation of the model is the calculation of the smoothed time-derivative of the log-compressed stimulus envelope. The ability of the model to reproduce cortical response patterns in detail across a variety of stimulus conditions suggests similar time-sensitive mechanisms may contribute to the physiological correlates of CMR.

These findings augment our previous observations that the simple edge-detection model can successfully predict a wide range of physiological and perceptual phenomena [11]. Former applications of the model to perceptual phenomena have been mainly related to auditory scene analysis, or more specifically the ability of the auditory system to distinguish multiple sound sources. In these cases, a sharp amplitude transition at stimulus onset ("auditory edge") was critical for sound segregation. Here, it is shown that the detection of acoustic signals also may be enhanced through the suppression of ongoing responses to the concurrent modulations of competing background sounds. Interestingly, these temporal fluctuations appear to be a common property of natural soundscapes [15].

The model provides testable predictions regarding how signal detection may be influenced by the temporal shape of amplitude modulation. Carlyon et al. [6] measured CMR in human listeners using three types of noise modulation: square-wave, sine wave and multiplied noise. From the perspective of the edge-detection model, these psychoacoustic results are intriguing because the different modulator types represent manipulations of the time derivative of masker envelopes. Square-wave modulation had the most sharply edged time derivative and produced the greatest masking release.

Fig. 5 plots the responses of the model to a pure-tone signal in square-wave and sine-wave modulated noise. As in the psychoacoustical data of Carlyon et al., the simulated detection threshold was lower in the context of square-wave modulation. Our modeling results suggest that the sharply edged square wave evoked higher levels of noise-driven activity and therefore created a sensitive background for the suppressing effects of the unmodulated tone.

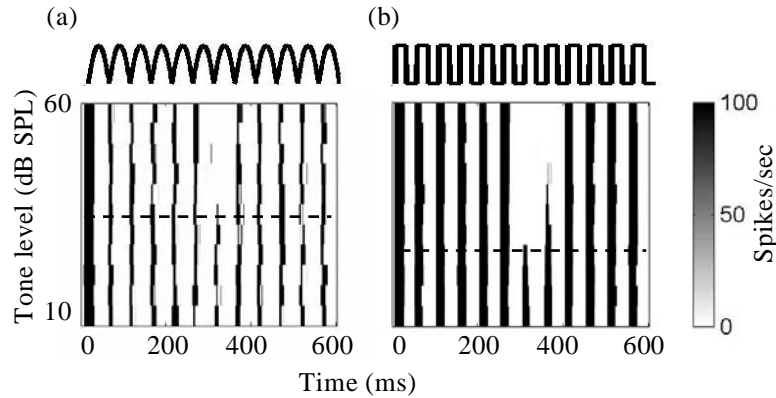

Figure 5: Simulated responses of the model to a combination of a tone at various levels and a sine-wave modulated noise (*a*) or a square-wave modulated noise (*b*). Each row shows the response of the model to the noise plus the tone at the level specified on the abscissa. The shape of the noise modulator is illustrated above each figure. The 100 ms tone starts 250 ms after the noise onset. Note that the tone detection threshold (marked by the dashed line) is 10 dB lower for the square-wave modulator than for the sine-wave modulator, in accordance with the psychoacoustical data of Carlyon et al. [6].

Although the physiological basis of our model was derived from studies of neural responses in the cat auditory system, the key psychoacoustical observations of Carlyon et al. have been replicated in recent behavioral studies of cats (Budelis et al. [5]).

These data support the generalization of human perceptual processing to other species and enhance the possible correspondence between the neuronal CMR-like effect and the psychoacoustical masking phenomena.

Clearly, the auditory system relies on information other than the time derivative of the stimulus envelope for the detection of auditory signals in background noise. Further physiological and psychoacoustic assessments of CMR-like masking effects are needed not only to refine the predictive abilities of the edge-detection model but also to reveal the additional sources of acoustic information that influence signal detection in constantly changing natural environments.

## Acknowledgments

This work was supported in part by a NIDCD grant R01 DC004841.

## References

[1] Agmon-Snir H., Segev I. (1993). "Signal delay and input synchronization in passive dendritic structure", *J. Neurophysiol.* 70, 2066-2085.

[2] Bregman A.S. (1990). "Auditory scene analysis: The perceptual organization of sound", MIT Press, Cambridge, MA.

[3] Bregman A.S., Ahad P.A., Kim J., Melnerich L. (1994) "Resetting the pitch-analysis system. 1. Effects of rise times of tones in noise backgrounds or of harmonics in a complex tone", *Percept. Psychophys.* 56 (2), 155-162.

[4] Bregman A.S., Ahad P.A., Kim J. (1994) "Resetting the pitch-analysis system. 2. Role of sudden onsets and offsets in the perception of individual components in a cluster of overlapping tones", *J. Acoust. Soc. Am.* 96 (5), 2694-2703.

[5] Budelis J., Fishbach A., May B.J. (2002) "Behavioral assessments of comodulation masking release in cats", Abst. Assoc. for Res. in Otolaryngol. 25.

[6] Carlyon R.P., Buus S., Florentine M. (1989) "Comodulation masking release for three types of modulator as a function of modulation rate", *Hear. Res.* 42, 37-46.

[7] Darwin C.J. (1997) "Auditory grouping", *Trends in Cog. Sci.* 1(9), 327-333.

[8] Darwin C.J., Ciocca V. (1992) "Grouping in pitch perception: Effects of onset asynchrony and ear of presentation of a mistuned component", *J. Acoust. Soc. Am.* 91 , 3381-3390.

[9] Drullman R., Festen H.M., Plomp R. (1994) "Effect of temporal envelope smearing on speech reception", *J. Acoust. Soc. Am.* 95 (2), 1053-1064.

[10] Eggermont J J. (1994). "Temporal modulation transfer functions for AM and FM stimuli in cat auditory cortex. Effects of carrier type, modulating waveform and intensity", *Hear. Res.* 74, 51-66.

[11] Fishbach A., Nelken I., Yeshurun Y. (2001) "Auditory edge detection: a neural model for physiological and psychoacoustical responses to amplitude transients", *J. Neurophysiol.* 85, 2303–2323.

[12] Gerstner W. (1999) "Spiking neurons", in *Pulsed Neural Networks*, edited by W. Maass, C. M. Bishop, (MIT Press, Cambridge, MA).

[13] Hall J.W., Haggard M.P., Fernandes M.A. (1984) "Detection in noise by spectro-temporal pattern analysis", *J. Acoust. Soc. Am.* 76, 50-56.

[14] Heil P. (1997) "Auditory onset responses revisited. II. Response strength", *J. Neurophysiol.* 77, 2642-2660.

[15] Nelken I., Rotman Y., Bar-Yosef O. (1999) "Responses of auditory cortex neurons to structural features of natural sounds", *Nature* 397, 154-157.

[16] Phillips D.P. (1988). "Effect of Tone-Pulse Rise Time on Rate-Level Functions of Cat Auditory Cortex Neurons: Excitatory and Inhibitory Processes Shaping Responses to Tone Onset", *J. Neurophysiol.* 59, 1524-1539.

[17] Phillips D.P., Burkard R. (1999). "Response magnitude and timing of auditory response initiation in the inferior colliculus of the awake chinchilla", *J. Acoust. Soc. Am.* 105, 2731-2737.

[18] Phillips D.P., Semple M.N., Kitzes L.M. (1995). "Factors shaping the tone level sensitivity of single neurons in posterior field of cat auditory cortex", *J. Neurophysiol.* 73, 674-686.

[19] Rosen S. (1992) "Temporal information in speech: acoustic, auditory and linguistic aspects", *Phil. Trans. R. Soc. Lond.* B 336, 367-373.

[20] Shannon R.V., Zeng F.G., Kamath V., Wygonski J, Ekelid M. (1995) "Speech recognition with primarily temporal cues", *Science* 270, 303-304.

[21] Turner C.W., Relkin E.M., Doucet J. (1994). "Psychophysical and physiological forward masking studies: probe duration and rise-time effects", *J. Acoust. Soc. Am.* 96 (2), 795-800.

[22] Yost W.A., Sheft S. (1994) "Modulation detection interference – across-frequency processing and auditory grouping", *Hear. Res.* 79, 48-58.

[23] Zhang X., Heinz M.G., Bruce I.C., Carney L.H. (2001). "A phenomenological model for the responses of auditory-nerve fibers: I. Nonlinear tuning with compression and suppression", *J. Acoust. Soc. Am.* 109 (2), 648-670.
